# Limits on Learning Machine Accuracy Imposed by Data Quality

**Corinna Cortes, L. D. Jackel, and Wan-Ping Chiang**
AT&T Bell Laboratories
Holmdel, NJ 07733

## Abstract

Random errors and insufficiencies in databases limit the performance of any classifier trained from and applied to the database. In this paper we propose a method to estimate the limiting performance of classifiers imposed by the database. We demonstrate this technique on the task of predicting failure in telecommunication paths.

## 1 Introduction

Data collection for a classification or regression task is prone to random errors, e.g. inaccuracies in the measurements of the input or mis-labeling of the output. Missing or insufficient data are other sources that may complicate a learning task and hinder accurate performance of the trained machine. These insufficiencies of the data limit the performance of any learning machine or other statistical tool constructed from and applied to the data collection — no matter how complex the machine or how much data is used to train it.

In this paper we propose a method for estimating the limiting performance of learning machines imposed by the quality of the database used for the task. The method involves a series of learning experiments. The extracted result is, however, independent of the choice of learning machine used for these experiments since the estimated limiting performance expresses a characteristic of the data. The only requirements on the learning machines are that their capacity (VC-dimension) can be varied and can be made large, and that the learning machines with increasing capacity become capable of implementing *any* function.

We have applied the technique to data collected for the purpose of predicting failures in telecommunication channels of the AT&T network. We extracted information from one of AT&T's large databases that continuously logs performance parameters of the network. The character and amount of data comes to more material than humans can survey. The processing of the extracted information is therefore automated by learning machines.

We conjecture that the quality of the data imposes a limiting error rate on any learning machine of $\sim 25\%$, so that even with an unlimited amount of data, and an arbitrarily complex learning machine, the performance for this task will not exceed $\sim 75\%$ correct. This conjecture is supported by experiments.

The relatively high noise-level of the data, which carries over to a poor performance of the trained classifier, is typical for many applications: the data collection was not designed for the task at hand and proved inadequate for constructing high performance classifiers.

## 2    Basic Concepts of Machine Learning

We can picture a learning machine as a device that takes an unknown input vector and produces an output value. More formally, it performs some mapping from an input space to an output space. The particular mapping it implements depends of the setting of the internal parameters of the learning machine. These parameters are adjusted during a learning phase so that the labels produced on the training set match, as well as possible, the labels provided. The number of patterns that the machine can match is loosely called the "capacity" of the machine. Generally, the capacity of a machine increases with the number of free parameters. After training is complete, the generalization ability of of the machine is estimated by its performance on a test set which the machine has never seen before.

The test and training error depend on both the the number of training examples $l$, the capacity $h$ of the machine, and, of course, how well suited the machine is to implement the task at hand. Let us first discuss the typical behavior of the test and training error for a noise corrupted task as we vary $h$ but keep the amount $l$ of training data fixed. This scenario can, e.g., be obtained by increasing the number of hidden units in a neural network or increasing the number of codebook vectors in a Learning Vector Quantization algorithm [6]. Figure 1a) shows typical training and test error as a function of the capacity of the learning machine. For $h \ll l$ we have many fewer free parameters than training examples and the machine is over constrained. It does not have enough complexity to model the regularities of the training data, so both the training and test error are large (underfitting). As we increase $h$ the machine can begin to fit the general trends in the data which carries over to the test set, so both error measures decline. Because the performance of the machine is optimized on only part of the full pattern space the test error will always be larger than the training error. As we continue to increase the capacity of the learning machine the error on the training set continues to decline, and eventually it reaches zero as we get enough free parameters to completely model the training set. The behavior of the error on the test set is different. Initially it decreases, but at some capacity, $h^\star$, it starts to rise. The rise occurs because the now ample resources of the training machine are applied to learning vagaries of the training

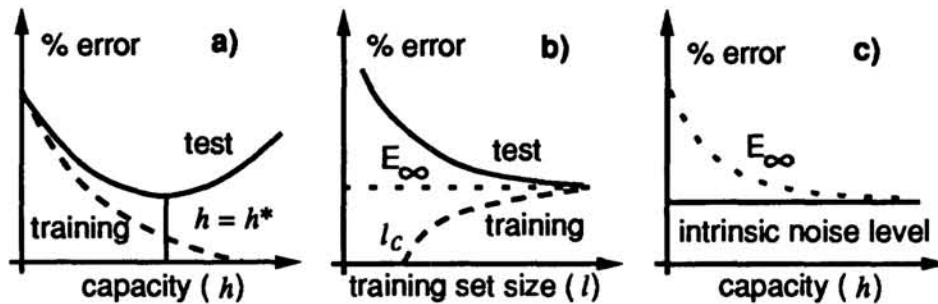

**Figure 1**: Errors as function of capacity and training set size. Figure 1a) shows characteristic plots of training and test error as a function of the learning machine capacity for fixed training set size. The test error reaches a minimum at $h = h^\star$ while the training error decreases as $h$ increases. Figure 1b) shows the training and test errors at fixed $h$ for varying $l$. The dotted line marks the asymptotic error $E_\infty$ for infinite $l$. Figure 1c) shows the asymptotic error as a function of $h$. This error is limited from below by the intrinsic noise in the data.

set, which are not reproduced in the test set (overfitting). Notice how in Figure 1a) the optimal test error is achieved at a capacity $h^\star$ that is smaller than the capacity for which zero error is achieved on the training set. The learning machine with capacity $h^\star$ will typically commit errors on misclassified or outlying patterns of the training set.

We can alternatively discuss the error on the test and training set as a function of the training set size $l$ for fixed capacity $h$ of the learning machine. Typical behavior is sketched in Figure 1b). For small $l$ we have enough free parameters to completely model the training set, so the training error is zero. Excess capacity is used by the learning machine to model details in the training set, leading to a large test error. As we increase the training set size $l$ we train on more and more patterns so the test error declines. For some critical size of the training set, $l_c$, the machine can no longer model all the training patterns and the training error starts to rise. As we further increase $l$ the irregularities of the individual training patterns smooth out and the parameters of the learning machine is more and more used to model the true underlying function. The test error declines, and asymptotically the training and test error reach the same error value $E_\infty$. This error value is the limiting performance of the given learning machine to the task. In practice we never have the infinite amount of training data needed to achieve $E_\infty$. However, recent theoretical calculations [8, 1, 2, 7, 5] and experimental results [3] have shown that we can estimate $E_\infty$ by averaging the training and test errors for $l > l_c$. This means we can predict the optimal performance of a given machine.

For a given type of learning machine the value of the asymptotic error $E_\infty$ of the machine depends on the quality of the data and the set of functions it can implement. The set of available functions increases with the capacity of the machine:

low capacity machines will typically exhibit a high asymptotic error due to a big difference between the true noise-free function of the patterns and the function implemented by the learning machine, but as we increase $h$ this difference decreases. If the learning machine with increasing $h$ becomes a *universal* machine capable of modeling *any* function the difference eventually reaches zero, so the asymptotic error $E_\infty$ only measures the intrinsic noise level of the data. Once a capacity of the machine has been reached that matches the complexity of the true function no further improvement in $E_\infty$ can be achieved. This is illustrated in Figure 1c). The intrinsic noise level of the data or the limiting performance of any learning machine may hence be estimated as the asymptotic value of $E_\infty$ as obtained for asymptotically universal learning machines with increasing capacity applied to the task. This technique will be illustrated in the following section.

## 3   Experimental Results

In this section we estimate the limiting performance imposed by the data of any learning machine applied to the particular prediction task.

### 3.1   Task Description

To ensure the highest possible quality of service, the performance parameters of the AT&T network are constantly monitored. Due to the high complexity of the network this performance surveillance is mainly corrective: when certain measures exceed preset thresholds action is taken to maintain reliable, high quality service. These reorganizations can lead to short, minor impairments of the quality of the communication path. In contrast, the work reported here is preventive: our objective is to make use of the performance parameters to form predictions that are sufficiently accurate that preemptive repairs of the channels can be made during periods of low traffic.

In our study we have examined the characteristics of long-distance, 45 Mbits/s communication paths in the domestic AT&T network. The paths are specified from one city to another and may include different kinds of physical links to complete the paths. A path from New York City to Los Angeles might include both optical fiber and coaxial cable. To maintain high-quality service, particular links in a path may be switched out and replaced by other, redundant links.

There are two primary ways in which performance degradation is manifested in the path. First is the simple bit-error rate, the fraction of transmitted bits that are not correctly received at the termination of the path. Barring catastrophic failure (like a cable being cut), this error rate can be measured by examining the error-checking bits that are transmitted along with the data. The second instance of degradation, "framing error", is the failure of synchronization between the transmitter and receiver in a path. A framing error implies a high count of errored bits.

In order to better characterize the distribution of bit errors, several measures are historically used to quantify the path performance in a 15 minutes interval. These measures are:

**Low-Rate** The number of seconds with exactly 1 error.

**"No–Trouble" patterns:**

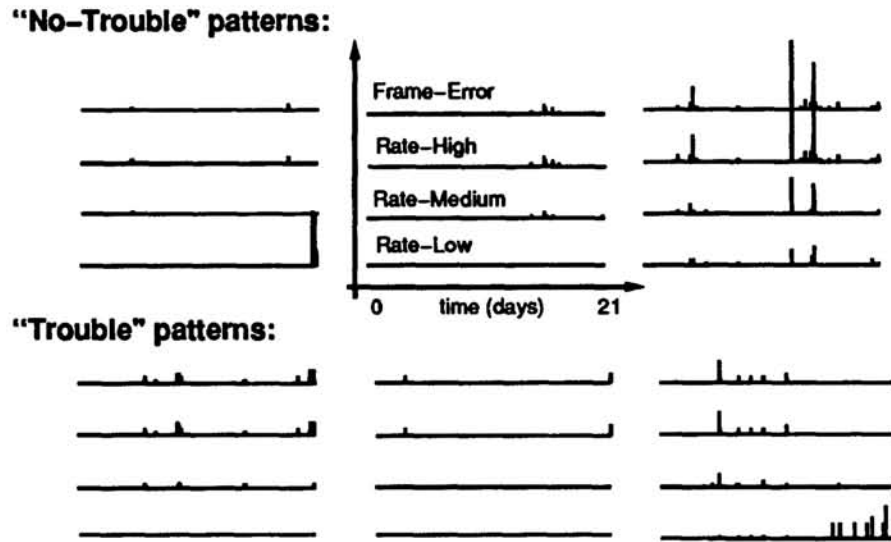

**"Trouble" patterns:**

**Figure 2**: Errors as function of time. The 3 top patterns are members of the "No-Trouble" class. The 3 bottom ones are members of the "Trouble" class. Errors are here plotted as mean values over hours.

**Medium-Rate** The number of seconds with more than one but less than 45 errors.

**High-Rate** The number of seconds with 45 or more errors, corresponding to a bit error rate of at least $10^{-6}$.

**Frame-Error** The number of seconds with a framing error. A second with a frame-error is always accompanied by a second of High-Rate error.

Although the number of seconds with the errors described above in principle could be as high as 900, any value greater than 255 is automatically clipped back to 255 so that each error measure value can be stored in 8 bits.

Daily data that include these measures are continuously logged in an AT&T database that we call Perf(ormance)Mon(itor). Since a channel is error free most of the time, an entry in the database is only made if its error measures for a 15 minute period exceed fixed low thresholds, e.g. 4 Low-Rate seconds, 1 Medium- or High-Rate second, or 1 Frame-Error. In our research we "mined" PerfMon to formulate a prediction strategy. We extracted examples of path histories 28 days long where the path at day 21 had at least 1 entry in the PerfMon database. We labeled the examples according to the error-measures over the next 7 days. If the channel exhibited a 15-minute period with at least 5 High-Rate seconds we labeled it as belonging to the class "Trouble". Otherwise we labeled it as member of "No-Trouble".

The length of the history- and future-windows are set somewhat arbitrarily. The history has to be long enough to capture the state of the path but short enough that our learning machine will run in a reasonable time. Also the longer the history the more likely the physical implementation of the path was modified so the error measures correspond to different media. Such error histories could in principle be eliminated from the extracted examples using the record of the repairs and changes

of the network. The complexity of this database, however, hinders this filtering of examples. The future-window of 7 days was set as a design criterion by the network system engineers.

Examples of histories drawn from PerfMon are shown in Figure 2. Each group of traces in the figure includes plots of the 4 error measures previously described. The 3 groups at the top are examples that resulted in No-Trouble while the examples at the bottom resulted in Trouble. Notice how bursty and irregular the errors are, and how the overall level of Frame- and High-Rate errors for the Trouble class seems only slightly higher than for the No-Trouble class, indicating the difficulty of the classification task as defined from the database PerfMon. PerfMon constitutes, however, the only stored information about the state of a given channel in its entirety and thus all the knowledge on which one can base channel end-to-end predictions: it is impossible to install extra monitoring equipment to provide other than the 4 mentioned end-to-end error measures.

The above criteria for constructing examples and labels for 3 months of PerfMon data resulted in 16325 examples from about 900 different paths with 33.2% of the examples in the class Trouble. This means, that always guessing the label of the largest class, No-Trouble, would produce an error rate of about 33%.

## 3.2   Estimating Limiting Performance

The 16325 path examples were randomly divided into a training set of 14512 examples and a test set of 1813 examples. Care was taken to ensure that a path only contributes to one of the sets so the two sets were independent, and that the two sets had similar statistical properties.

Our input data has a time-resolution of 15 minutes. For the results reported here the 4 error measures of the patterns were subsampled to mean values over days yielding an input dimensionality of $4 \times 21$.

We performed two sets of independent experiments. In one experiment we used fully connected neural networks with one layer of hidden units. In the other we used LVQ learning machines with an increasing number of codebook vectors. Both choices of machine have two advantages: the capacity of the machine can easily be increased by adding more hidden units, and by increasing the number of hidden units or number of codebook vectors we can eventually model *any* mapping [4]. We first discuss the results with neural networks.

Baseline performance was obtained from a threshold classifier by averaging all the input signals and thresholding the result. The training data was used to adjust the single threshold parameter. With this classifier we obtained 32% error on the training set and 33% error on the test set. The small difference between the two error measures indicate statistically induced differences in the difficulty of the training and test sets. An analysis of the errors committed revealed that the performance of this classifier is almost identical to always guessing the label of the largest class "No-Trouble": close to 100% of the errors are false negative.

A linear classifier with about 200 weights (the network has two output units) obtained 28% error on the training set and 32% error on the test set.

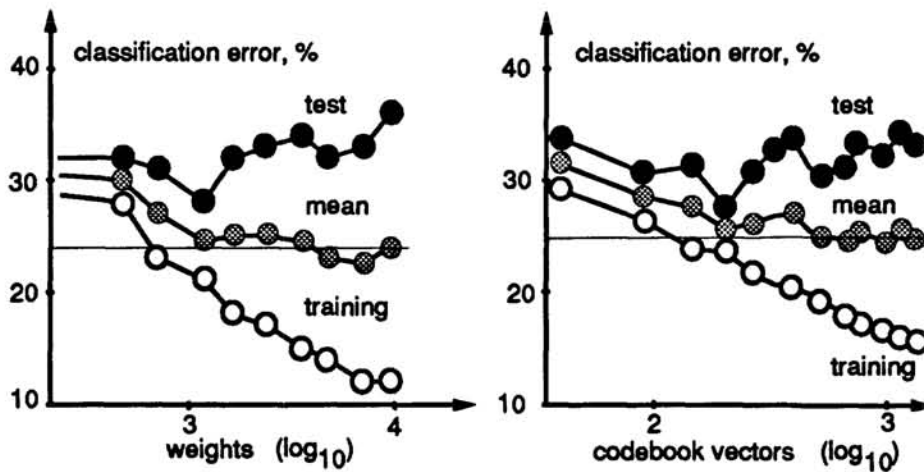

**Figure 3**: a) Measured classification errors for neural networks with increasing number of weights (capacity). The mean value between the test and training error estimates the performance of the given classifier trained with unlimited data. b) Measured classification errors for LVQ classifiers with increasing number of codebook vectors.

Further experiments exploited neural nets with one layer of respectively 3, 5, 7, 10, 15, 20, 30, and 40 hidden units. All our results are summarized in Figure 3a). This figure illustrates several points mentioned in the text above. As the complexity of the network increases, the training error decreases because the networks get more free parameters to memorize the data. Compare to Figure 1a). The test error also decreases at first, going through a minimum of 29% at the network with 5 hidden units. This network apparently has a capacity that best matches the amount and character of the available training data. For higher capacity the networks overfit the data at the expense of increased error on the test set.

Figure 3a) should also be compared to Figure 1c). In Figure 3a) we plotted approximate values of $E_\infty$ for the various networks — the minimal error of the network to the given task. The values of $E_\infty$ are estimated as the mean of the training and test errors. The value of $E_\infty$ appears to flatten out around the network with 30 units, asymptotically reaching a value of 24% error.

An asymptotic $E_\infty$-value of 25% was obtained from LVQ-experiments with increasing number of codebook vectors. These results are summarized in Figure 3b). We therefore conjecture that the intrinsic noise level of the task is about 25%, and this number is the limiting error rate imposed by the quality of the data on any learning machine applied to the task.

## 4   Conclusion

In this paper we have proposed a method for estimating the limits on performance imposed by the quality of the database on which a task is defined. The method involves a series of learning experiments. The extracted result is, however, independent of the choice of learning machine used for these experiments since the estimated limiting performance expresses a characteristic of the data. The only requirements on the learning machines are that their capacity can be varied and be made large, and that the machines with increasing capacity become capable of implementing *any* function. In this paper we have demonstrated the robustness of our method to the choice of classifiers: the result obtained with neural networks is in statistical agreement with the result obtained for LVQ classifiers.

Using the proposed method we have investigated how well prediction of upcoming trouble in a telecommunication path can be performed based on information extracted from a given database. The analysis has revealed a very high intrinsic noise level of the extracted information and demonstrated the inadequacy of the data to construct high performance classifiers. This study is typical for many applications where the data collection was not necessarily designed for the problem at hand.

**Acknowledgments**

We gratefully acknowledge Vladimir Vapnik who brought this application to the attention of the Holmdel authors. One of the authors (CC) would also like to thank Walter Dziama, Charlene Paul, Susan Blackwood, Eric Noel, and Harris Drucker for lengthy explanations and helpful discussions of the AT&T transport system.

## References

[1] S. Bös, W. Kinzel, and M. Opper. The generalization ability of perceptrons with continuous output. *Physical Review E*, 47:1384–1391, 1993.

[2] Corinna Cortes. *Prediction of Generalization Ability in Learning Machines.* PhD thesis, University of Rochester, NY, 1993.

[3] Corinna Cortes, L. D. Jackel, Sara A. Solla, V. Vapnik, and John S. Denker. Learning curves: Asymptotic value and rate of convergence. In *Advances in Neural Information Processing Systems*, volume 6. Morgan Kaufman, 1994.

[4] G. Cybenko, K. Hornik, M. Stinchomb, and H. White. Multilayer feedforward neural networks are universal approximators. *Neural Networks*, 2:359–366, 1989.

[5] T. L. Fine. Statistical generalization and learning. Technical Report EE577, Cornell University, 1993.

[6] Teuvo Kohonen, György Barna, and Ronald Chrisley. Statistical pattern recognition with neural networks: Benchmarking studies. In *Proc. IEEE Int. Conf. on Neural Networks, IJCNN-88*, volume 1, pages I-61—I-68, 1988.

[7] N. Murata, S. Yoshizawa, and S. Amari. Learning curves, model selection, and complexity of neural networks. In *Advances in Neural Information Processing Systems*, volume 5, pages 607–614. Morgan Kaufman, 1992.

[8] H. S. Seung, H. Sompolinsky, and N. Tishby. Statistical mechanics of learning from examples. *Physical Review A*, 45:6056–6091, 1992.
